# Learning a Forward Model of a Reflex

**Bernd Porr and Florentin Wörgötter**
Computational Neuroscience
Psychology
University of Stirling
FK9 4LR Stirling, UK
{bp1,faw1}@cn.stir.ac.uk

## Abstract

We develop a systems theoretical treatment of a behavioural system that interacts with its environment in a closed loop situation such that its motor actions influence its sensor inputs. The simplest form of a feedback is a reflex. Reflexes occur always "too late"; i.e., only *after* a (unpleasant, painful, dangerous) reflex-eliciting sensor event has occurred. This defines an objective problem which can be solved if another sensor input exists which can predict the primary reflex and can generate an earlier reaction. In contrast to previous approaches, our linear learning algorithm allows for an analytical proof that this system learns to apply feedforward control with the result that slow feedback loops are replaced by their equivalent feed-forward controller creating a forward model. In other words, learning turns the reactive system into a pro-active system. By means of a robot implementation we demonstrate the applicability of the theoretical results which can be used in a variety of different areas in physics and engineering.

## 1   Introduction

Feedback loops are prevalent in animal behaviour, where they are normally called a "reflex". However, the reflex has the disadvantage of always being too late. Thus, an objective goal is to avoid a reflex (feedback) reaction. This can be done by an anticipatory (feedforward) action; for example when retracting a limb in response to heat radiation without actually having to touch the hot surface, which would elicit a pain-induced reflex. While this has been interpreted as successful forward control [1] the question arises how such a behavioural system can be robustly generated.

In this article we introduce a linear algorithm for temporal sequence learning between two sensor events and provide an analytical proof that this process turns a pre-wired reflex loop into its equivalent feed-forward controller. After learning the system will respond with an anticipatory action thereby avoiding the reflex.

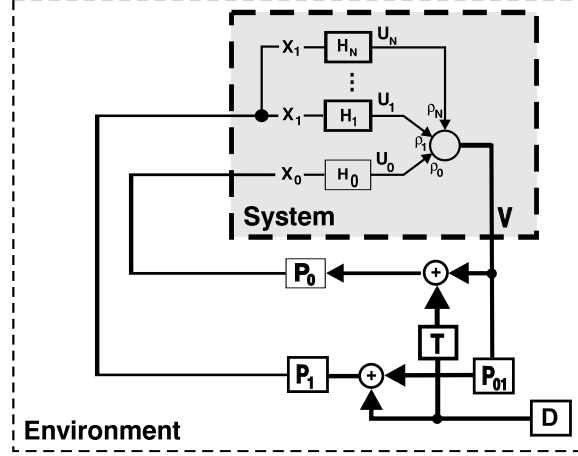

Figure 1: Diagram of the system in its environment (in Laplace-notation). The input signal is $D$ ("disturbance") reaching both sensor inputs $X_{0,1}$ at different times as indicated by the temporal delay $T$. The environmental transfer functions are denoted as $P$. $H$ are linear transfer functions, $U$ the filtered inputs which converge with weights $\rho$ onto the output neuron $V$.

## 2   The learning rule and its environment

Fig. 1 shows the general situation which arises when temporal sequence learning takes place in a system which interacts with its environment [2]. We distinguish two loops: The inner loop represents the reflex which has fixed unchanging properties. The outer loop represents the to-be-learned anticipatory action. Sequence learning requires causally related input events $D$ at both sensors $X_{0,1}$ (e.g. heat radiation and pain) where $T$ denotes the time delay between both inputs. The outer loop receives the earlier (anticipatory) input. The delayed and un-delayed signals $X_{0,1}$ are processed by a linear transform $H$ (e.g. a low- or band-pass filter), subsequently their sum is taken with weights $\rho$ on a single neuron. Note that *all* input signals are filtered. The system is therefore completely isotropic. Line $X_1$ is fanned out in order to adjust to the a priori unknown delay $T$ by the combination of different transforms $H_j$ (see below). The output of the neuron is in the LAPLACE-domain given by:

$$V(s) = \rho_0 U_0(s) + \sum_{j=1}^{N} \rho_j U_j(s), \qquad \text{with} \qquad U(s) = X(s)H(s) \qquad (1)$$

where $\rho_j$ are the synaptic weights. In the following we will drop the function argument $s$ for the sake of brevity wherever possible. The transfer functions $P$ in Fig. 1 denote how the environment influences the different signals. The goal of sequence learning is that the outer loop should after learning functionally replace the inner loop such that the reflex will cease to be triggered. In this case we receive $X_0 = 0$ which we call the "desired state" of the system. This allows calculating the general requirements for the outer loop without having to specify the actual learning process. The reflex pathway is described by

$$X_0 = P_0 [V + De^{-sT}] , \qquad (2)$$

where $e^{-sT}$ represents the delay $T$ in LAPLACE-notation. The signal on the anticipatory (outer) pathway has the representation

$$X_1 = \frac{P_1 D + P_1 P_{01} X_0 H_0}{1 - P_1 P_{01} H_v} \qquad (3)$$

where $H_v = \sum_{k=1}^{N} \rho_k H_k$ is the learned transfer-function which generates the anticipatory response triggered by the input $x_1$. We want to express $H_V$ by the environmental transfer-functions $P_0, P_1$ and $P_{01}$. $H_V$ is solved for the condition $X_0 = 0$ where the reflex is no longer triggered. Eliminating $X_1$ and $V$ we get:

$$H_v = -\frac{P_1^{-1}e^{-sT}}{1 - P_{01}e^{-sT}} \tag{4}$$

Eq. 4 can be further simplified. Following standard control theory [3] we neglect the denominator, because it does not add additional poles to the transfer function $H_v$. Such a pole appears only for $P_{01} = e^{sT}$. A transfer function $e^{sT}$, however, is meaningless because it violates temporal causality. Thus, the denominator can at most add phase-shifts to the systems behaviour. As a consequence, we may set $P_{01} := 0$ and the behaviour of $H_v(s)$ is determined by:

$$H_v(s) = -P_1^{-1}e^{-sT} \tag{5}$$

The interpretation of the last equation is straight-forward. The learning goal of $X_0 = 0$ requires compensating the disturbance $D$. The disturbance, however, enters the system only after having been filtered by the environmental transfer function $P_1$. Thus, compensation of $D$ requires to reverse this filtering by a term $P_1^{-1}$ which is the inverse environmental transfer function (hence "inverse controller"). The second term $e^{-sT}$ in Eq. 5 compensates for the delay $T$ between the two sensor signals originating from the disturbance $D$.

Having outlined the general setup in terms of our linear approach and system theoretic notation we devote the remaining three sections to the following topics: 2.1. The learning rule and convergence to a given solution $H_v$ under this rule. 2.2. The construction of (approximate) solutions $H_v$. 3. Implementation of the system in a (real world) robot experiment.

## 2.1 The learning rule and convergence.

Here, we assume that a set of functions $H$ exists (as will be be specified below) for which a solution can be approximated by $H_v = \sum_{j=1}^{N} \rho_j H_j$. We will now specify the learning rule, by which the development of the weight values is controlled and show that any deviation from the given solution $H_v$ is eliminated due to learning. In terms of the time domain functions $u_j, v$ corresponding to $U_j$ and $V$, our learning rule is given by:

$$\frac{d}{dt}\rho_j = \mu u_j v' \qquad \mu \ll 1 \tag{6}$$

Thus, the weight change depends on the correlation between $u_j$ and the time derivative of $v$. Since the structure of the system is completely isotropic (see Fig. 1) and learning can take place at any synapse we shall call our learning algorithm isotropic sequence order learning ("ISO-learning"). The positive constant $\mu$ is taken small enough such that all weight changes occur on a much longer time scale (i.e., very slowly) as compared to the decay of the responses $u$. This rule is related to the one used in "temporal difference" learning [4]. The total weight change can be calculated by [5]:

$$\Delta\rho_j = \frac{\mu}{2\pi} \int_{-\infty}^{\infty} -i\omega V(-i\omega)U_j(i\omega)d\omega \tag{7}$$

where $-i\omega V(-i\omega)$ represents the derivative of $v$ in the LAPLACE domain. We assume that the reflex pathway is unchanging with a fixed weight $\rho_0 < 0$ (negative feedback). Note, that its open loop transfer characteristic given by $\rho_0 H_0 P_0$ must carry a low-pass component, otherwise the reflex loop would be unstable. We keep $P_{01} = 0$ as before. Furthermore we assume that for a given set of $H_j$ we have found a set of weights $\rho_j, 1 \le j \le N$ which solves Eq. 5. We will show that a perturbation of the weights $\rho_j$ will be

compensated by applying the learning procedure. Since we do not make any assumption as to the size of the perturbation this is indicative of convergence in general. To this end, we substitute $\rho_j \to \rho_j + \delta\rho_j$. Stability of the solution is expected if the weight change $\Delta\rho_j$ opposes the perturbation, thus, if $\Delta\rho_j \sim (-1)\delta\rho_j$. Here, we however assume an 'adiabatic' environment in which the system internally relaxes on a time scale much shorter than the time scale on which the disturbances occur. To be specific, a disturbance/perturbation may occur near $t = 0$. In calculating the weight change (7) due to this disturbance signal we disregard any subsequent disturbances as well as perturbations ($\delta\rho_j$) following the steady state condition. We use the relations for $U$ and $V$ and insert them into Eq. 7. For $U$ we have:

$$U_j = X_j H_j = \left\{ \begin{array}{ll} X_0 H_0 & for \quad j = 0 \\ X_1 H_j & for \quad j > 0 \end{array} \right. \tag{8}$$

Inserting Eqs. 2 and 8 into Eq. 1 we get:

$$V = \frac{\rho_0 P_0 H_0 D e^{-sT} + X_1 \sum_{j=1}^{N} \rho_j H_j}{1 - \rho_0 P_0 H_0} \tag{9}$$

Substituting $\rho_j \to \rho_j + \delta\rho_j$ this yields:

$$\tilde{V} = V + \frac{X_1 \sum_{j=1}^{N} \delta\rho_j H_j}{1 - \rho_0 P_0 H_0} \tag{10}$$

We use the superscript $^-$ and $^+$ to denote the arguments $(-i\omega)$ and $(i\omega)$ respectively and calculate the weight change using Eq. 7 integrating between $-\infty$ and $\infty$:

$$\Delta\rho_j = \frac{\mu}{2\pi} \int -i\omega [V^- + \frac{X_1^- \sum_{k=1}^{N} \delta\rho_k H_k^-}{1 - \rho_0 P_0^- H_0^-}] X_1^+ H_j^+ d\omega \tag{11}$$

We realize that the first part of this integral describes the unperturbed equilibrium state and can be dropped, thus, together with $X_1^+ X_1^- = |X_1|^2$, which holds because $X_1$ is a transfer function, we get:

$$\Delta\rho_j = \frac{\mu}{2\pi} \sum_{k=1}^{N} \delta\rho_k \int -i\omega \frac{|X_1^+|^2 H_j^+ H_k^-}{1 - \rho_0 P_0^- H_0^-} d\omega \tag{12}$$

Furthermore we assume orthogonality (see also below) given by:

$$0 = \int -i\omega \frac{|X_1^+|^2 H_j^+ H_k^-}{1 - \rho_0 P_0^- H_0^-} d\omega \quad \text{for } k \neq j \tag{13}$$

and get accordingly:

$$\Delta\rho_j = \frac{\mu}{2\pi} \delta\rho_j \int -i\omega \frac{|X_1^+|^2 |H_j^+|^2}{1 - \rho_0 P_0^- H_0^-} d\omega \tag{14}$$

$$= \frac{\mu}{2\pi} \delta\rho_j \int |X_1^+ H_j^+|^2 (-i\omega \frac{1}{1 - \rho_0 P_0^- H_0^-}) \tag{15}$$

We now apply PLANCHEREL'S theorem [5] in order to transfer the integral into the time-domain and prove that it is negative. This assures stability and, hence, convergence, because we know that $\mu$ is small, preventing oscillatory behaviour. We have:

$$\Delta\rho_j = \mu\delta\rho_j \int_0^{\infty} a_{x*h}(t) f'(t) dt \tag{16}$$

where we call $a_{x*h}(t)$ the autocorrelation function of $x_1(t) * h_j(t)$ which is the inverse transform of $|X_1^+ H_j^+|^2$ ($*$ denotes a convolution) and $f'(t)$ is the temporal derivative of the impulse response of the inverse transform of the remaining second term in Eq. 15. Since we know that $\rho_0 H_0 P_0$ must carry a low-pass component we can in general state that the fraction $\frac{1}{1-\rho_0 P_0 H_0}$ represents a (non-standard) high-pass. Its derivative has a very high negative value for $t = 0$ (ideally $= -\infty$) and vanishes soon thereafter. The autocorrelation $a$ is positive around $t = 0$. Thus, the integral in question will remain negative for almost all realistic choices of $x_1(t)$. As an important special case we find that this especially holds if we assume delta-pulse disturbance at $t = 0$, corresponding to $x_1(t) = \delta(t)$.

## 2.2 Construction of solutions.

Here, we use a set of well-known functions $H$ (band-pass filters) and show explicitly that a solution which approximates the inverse controller (Eq. 5) can be constructed for $N = 1$ and discuss how the approximation is improved for higher values of $N$.

The transfer functions of the band-pass filters $H$, which we use, are specified in the LAPLACE-domain: $H(s) = \frac{1}{(s+p)(s+p^*)}$ where $p^*$ represents the complex conjugate of the pole $p = a + ib$. Real and imaginary parts of the poles are given by $a = \mathrm{Re}(p) = -\pi f/Q, b = \mathrm{Im}(p) = \sqrt{(2\pi f)^2 - a^2}$, where $f$ is the frequency of the oscillation. The damping characteristic of the resonator is reflected by $Q > 0.5$. Concerning convergence one finds in Eq. 16 that with such a set of functions $f'(t) << 0$ for $t = 0$ and that $f'$ converges fast to zero for $t > 0$. Band-pass functions are not orthogonal to each other but numerically we found that they can be approximately treated of being orthogonal. In fact only a small drift of the weights is observed which could be compensated if required. In practise, however, this becomes unimportant as discussed below. The use of resonators is also motivated by biology [6] and band-pass filtered response characteristics are prevalent in neuronal systems which also have been used in other neuro-theoretical approaches [7].

We return to Eq. 5. Let us first assume that the environment does not filter the disturbance, thus $P_1 := 1$. Then, for the case $N = 1$, an approximative solution of Eq. 5 can be easily constructed by developing $-e^{-sT}$ into a Taylor series and obtaining the parameters through comparing coefficients in:

$$
\frac{1}{e^{sT}} = \frac{1}{1 + sT + \frac{1}{2}s^2T^2 + \ldots} \approx \frac{2T^{-2}}{2T^{-2} + 2sT^{-1} + s^2}
$$

$$
:= \frac{-\rho_1}{\underbrace{pp^*}_{(2\pi f_1)^2} + s\underbrace{(p + p^*)}_{-\frac{2\pi f_1}{Q_1}} + s^2} = -\rho_1 H_1(s) \tag{17}
$$

Accordingly we get for the parameters of $H_1$: $\rho_1 = -2\frac{1}{T^2}$, $f_1 = \pm\frac{1}{\pi\sqrt{2}}\frac{1}{T}$, $Q_1 = \sqrt{\frac{1}{2}}$. For un-filtered throughput $P_1 = 1$, this result shows that for all $T$ there exists a resonator $H_1$ with a weight $\rho_1$, which approximates $-e^{-sT}$ to the second order. The approximation continues to improve for higher orders of $N$, which we pursued up to $N = 2$ (fourth order Taylor), but the set of equations becomes rather cluttered. In general $P_1$ represents an environmental transfer function which is passive and "well-behaved". Thus, in most cases it can be represented by just another passive low- or band-pass filter (sum of complex conjugated poles). Under this assumption a solution can also be constructed for the complete term $-P_1^{-1}e^{-sT}$ by a combination of $N > 1$ resonators.

As mentioned above, constructing solutions becomes impractical for $N \neq 1$ and it would require to know $T$ and $P_1^{-1}$ a priori. Note, if you would know $P_1^{-1}$, you had already reached your goal of designing the inverse controller and learning would be obsolete. Thus, normally a set of resonators $H$ must be predefined in a somewhat arbitrary way and their weights $\rho$ shall be learned. The uniqueness of the solution assured by orthogonality becomes secondary in practise, because – without prior knowledge of $T$ and $P_1^{-1}$ – one has to use an over-complete set of $H$, in order to make sure that a solution can be found. In practise, this means that a large enough set of filters must be used which normally leads to a manifold of solutions. Now obviously the question arises if satisfactory solutions exist under these relaxed conditions and if they remain stable.

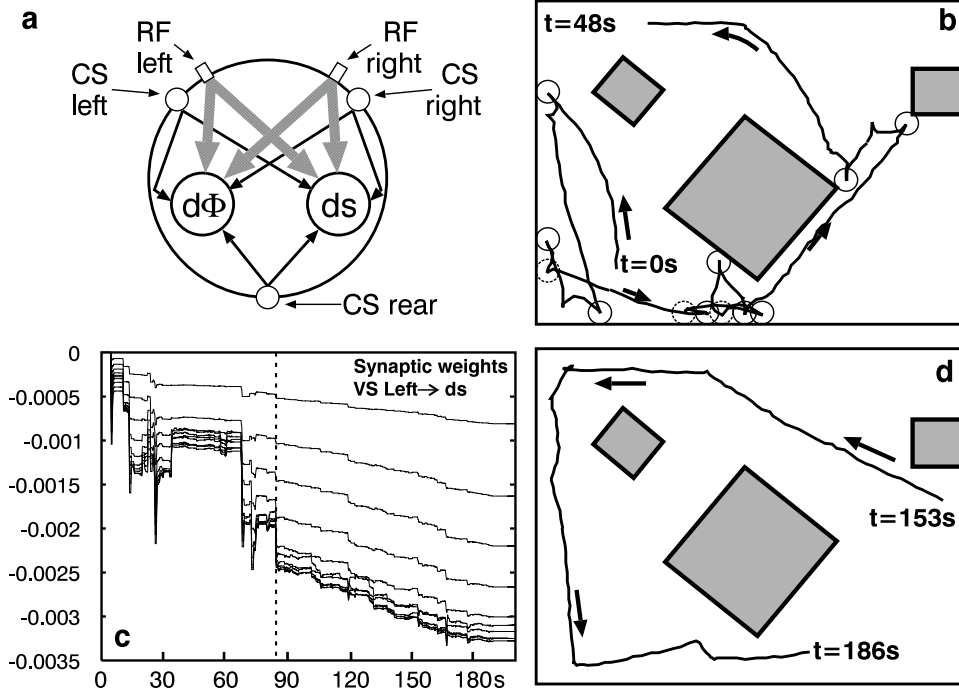

Figure 2: Robot experiment: (a) The robot has 2 output neurons for speed ($ds$) and steering angle ($d\phi$). The retraction mechanism is implemented by 3 resonators ($Q = 0.6$, $f = 1$Hz) which connect the collision sensors (CS) to the neurons $ds$ (speed) and $d\phi$ (steering angle) with fixed weights (reflex). Each range finder (RF) is fed into a filter bank of 10 resonators $H_k$ with $Q = 1.0$; $f_k = 10/k$Hz where its output converges with variable weights on both the $ds$ and $d\phi$-neuron. A more detailed technical description together with a set of movies can be found at: http://www.cn.stir.ac.uk/predictor/real – movie 1. (b,d) Parts of the motion trajectory for one trial in an arena of $240x200\ cm^2$ with three obstacles (shaded). Circles denote collisions. (c) Development of the weights from the left range finder sensor to the the neuron $ds$.

## 3 Implementation in a robot experiment.

In this section, we show a robot experiment where we apply a conventional filter bank approach using rather few filters with constant $Q$ and logarithmically spaced frequencies $f_k$ and demonstrate that the algorithms still produces the desired behaviour.

The task in this robot experiment is collision avoidance [8]. The built-in reflex-behaviour is a retraction reaction after the robot has hit an obstacle which represents the inner loop feedback mechanism[1]. The robot has three collision sensors ($X_0$) and two range finders ($X_1$), which produce the predictive signals. When driving around there is always a causal relation between the earlier occurring range finder signals and the later occurring collision, which drives the learning process. Fig. 2b shows that early during learning many collisions (circles) occur. After a collision a fast reflex-like retraction&turning reaction is elicited. On the other hand, the robot movement trace is now free of collisions after successful learning of the temporal correlation between range finder and collision signals (Fig. 2d) and the

trajectory is maximally smooth. The robot always found a stable solution, but those were - as expected - not unique. This is partly due to the different initial conditions but also due to the over-complete set of $H$. Possible solutions, which we have observed, are that the robot after learning simply stops in front of an obstacle and that it slightly oscillates back and forth. The more common solution of the robot is that it continuously drives around and uses mainly his steering to avoid obstacles. Note that this rather complex behaviour is established by only two neurons. Fig. 2c shows that the weight change slows down after the last collision has happened (dotted line in c). The still existing smaller weight change is due to the fact that after functional silencing of $X_0$ (no more collisions) temporally correlated inputs still exist namely *between* the left and right range finders. Thus, learning is now governed by these correlations instead and is driven by the earliest response of one of them which finally leads to the desired stabilisation.

## 4 Discussion

Replacing a feedback loop with its equivalent feed-forward controller is of central relevance for efficient control particularly in slow feedback systems, where long loop-delays exist. So far, feed-forward control is in general model-based and, thus, often not robust [9]. On the other hand, it has been suggested earlier by studies of limb movement control that temporal sequence learning could be used to solve the inverse controller problem [1].

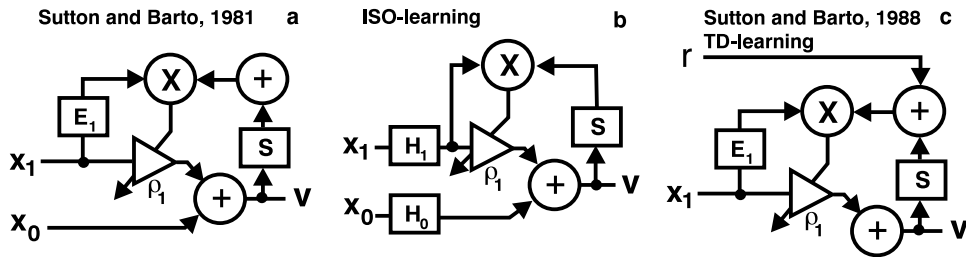

Figure 3: Differences between the Sutton and Barto models (a,c) and ISO-learning (b) in the case of $N = 1$. a) shows the drive reinforcement-model by Sutton and Barto [4] and c) the temporal difference (TD) learning by Sutton and Barto [10]. Note that the obsolete summation-point $\oplus$ in a) allows to add the reward-signal in c). b) shows ISO-learning like in Fig. 1 with $N = 1$. Additionally the circuit for the weight change (learning) is shown. The input-filters $E_1$ in the Sutton and Barto-models (a,c) are first order low-pass filters (eligibility trace). $\oplus$ and $\times$ represent addition and multiplication, respectively. $s$ is the derivative.

Widely used models of derivative based temporal sequence learning are those by Sutton and Barto which have the aim to model experiments of classical conditioning [4, 11, 10]. Fig. 3 shows their models in comparison to ISO-learning. All models strengthen the weight $\rho_1$ if $x_1$ precedes $x_0$ (or $r$, respectively). All models use filters at the inputs. However, in the Sutton and Barto-models these filtered input signals are only used as an input for the learning circuit (Fig. 3a,c) whereas the output is a superposition of the *original* input signals. Learning is therefore achieved by correlating the filtered input with the derivative of the (un-filtered) output-signal. Thus, filtered signals are correlated with un-filtered signals. In contrast to the Sutton and Barto-models, our model is completely isotropic and uses the filtered signals for *both*, the learning circuit and the output since the filtered signals are also responsible for an appropriate *behaviour* of the organism. These different wirings reflect the different learning goals: in our model the weight $\rho_1$ stabilises when the *input* $x_0$ has become silent (the reflex has been avoided). In the Sutton and Barto-models the

weight stabilises if the *output* has reached a specific condition. In the drive-reinforcement model this is the case if the output-signal $v$ caused by $x_1$ has a similar strength than the output $v$ triggered by $x_0$. This reflects the Rescorla/Wagner rule [12]. In the case of TD-learning learning stops if the prediction error between reward and the output $v$ is zero, thus if $v$ optimally predicts $r$. In general our model is closely related to any *correlation-based* sequence-learning [4, 13] and is *not* related to any form of reinforcement-learning [10, 14] as it does not need a special reward- or punishment-signal.

The current study demonstrates analytically the convergence of ISO-learning in a closed loop paradigm in conjunction with some rather general assumptions concerning the structure of such a system. Thus, this type of learning is able to generate a model-free inverse controller of a reflex, which improves the performance of conventional feedback-control, while the feedback still serves as a fall-back. Apart from biological implications this promises a broad field of applications in physics and engineering.

## Footnotes

[1]In fact it is also possible to construct an attraction-case if the reflex performs an initial attraction-reaction.

## References

[1] Daniel M. Wolpert and Zoubin Ghahramani. Computational principles of movement neuroscience. *Nature Neuroscience supplement*, 3:1212–1217, 2000.

[2] P. Read Montague, Peter Dayan, and Terrence J. Sejnowski. Bee foraging in uncertain environments using predictive hebbian learning. *Nature*, 377:725–728, 1995.

[3] W.E Sollecito and S.G Reque. Stability. In Jerry Fitzgerald, editor, *Fundamentals of System Analysis*, chapter 21. Wiley, New York, 1981.

[4] R.S. Sutton and A.G. Barto. Towards a modern theory of adaptive networks: expectation and prediction. *Psychol. Review*, 88:135–170, 1981.

[5] John L. Stewart. *Fundamentals of signal theory*. Mc Graw-Hill, New York, 1960.

[6] Gordon M. Shepherd, editor. *The synaptic organisation of the brain*. Oxford University Press, New York, 1990.

[7] Steven Grossberg. A spectral network model of pitch perception. *J Acoust Soc Am*, 98(2):862–879, 1995.

[8] P.F.M.J Verschure and T. Voegtlin. A bottom-up approach towards the aquisition, retention, and expression of sequential representations: Distributed adaptive control III. *Neural Networks*, 11:1531–1549, 1998.

[9] William J. Palm. *Modeling, Analysis and Control of Dynamic Systems*. Wiley, New York, 2000.

[10] R.S. Sutton. Learning to predict by method of temporal differences. *Machine learning*, 3(1):9–44, 1988.

[11] R.S. Sutton and A.G. Barto. Simulation of anticipatory responses in classical conditioning by a neuron-like adaptive element. *Behav. Brain. Res.*, 4(3):221–235, 1982.

[12] R.A. Rescorla and A.R. Wagner. A theory of pavlovian conditioning: Variations in the effectiveness of reinforcement and nonreinforcement. In A.H Black and W.F. Prokasy, editors, *Classical conditioning 2, current theory and research*, pages 64–99. ACC, New York, 1972.

[13] A. Harry Klopf. A drive-reinforcement model of single neuron function. In John S. Denker, editor, *Neural Networks for computing: AIP conference proceedings*, volume 151 of *AIP conference proceedings*, New York, 1986. American Institute of Physics.

[14] Christofer J.C.H Watkins and Peter Dayan. Q-learning. *Machine Learning*, 8:279–292, 1992.
